# PSDBoost: Matrix-Generation Linear Programming for Positive Semidefinite Matrices Learning

**Chunhua Shen**[†‡]**, Alan Welsh**[‡]**, Lei Wang**[‡]
[†]NICTA Canberra Research Lab, Canberra, ACT 2601, Australia[*]
[‡]Australian National University, Canberra, ACT 0200, Australia

## Abstract

In this work, we consider the problem of learning a positive semidefinite matrix. The critical issue is how to preserve positive semidefiniteness during the course of learning. Our algorithm is mainly inspired by LPBoost [1] and the general greedy convex optimization framework of Zhang [2]. We demonstrate the essence of the algorithm, termed PSDBoost (positive semidefinite Boosting), by focusing on a few different applications in machine learning. The proposed PSDBoost algorithm extends traditional Boosting algorithms in that its parameter is a positive semidefinite matrix with trace being one instead of a classifier. PSDBoost is based on the observation that any trace-one positive semidefinite matrix can be decomposed into linear convex combinations of trace-one rank-one matrices, which serve as base learners of PSDBoost. Numerical experiments are presented.

## 1    Introduction

Column generation (CG) [3] is a technique widely used in linear programming (LP) for solving large-sized problems. Thus far it has mainly been applied to solve problems with linear constraints. The proposed work here—which we dub matrix generation (MG)—extends the column generation technique to non-polyhedral semidefinite constraints. In particular, as an application we show how to use it for solving a semidefinite metric learning problem. The fundamental idea is to rephrase a bounded semidefinite constraint into a polyhedral one with infinitely many variables. This construction opens possibilities for use of the highly developed linear programming technology. Given the limitations of current semidefinite programming (SDP) solvers to deal with large-scale problems, the work presented here is of importance for many real applications.

The choice of a metric has a direct effect on the performance of many algorithms such as the simplest $k$-NN classifier and some clustering algorithms. Much effort has been spent on learning a good metric for pattern recognition and data mining. Clearly a good metric is task-dependent: different applications should use different measures for (dis)similarity between objects. We show how a Mahalanobis metric is learned from examples of proximity comparison among triples of training data. For example, assuming that we are given triples of images $\mathbf{a}_i$, $\mathbf{a}_j$ and $\mathbf{a}_k$ ($\mathbf{a}_i$, $\mathbf{a}_j$ have same labels and $\mathbf{a}_i$, $\mathbf{a}_k$ have different labels, $\mathbf{a}_i \in \mathbb{R}^D$), we want to learn a metric between pairs of images such that the distance from $\mathbf{a}_j$ to $\mathbf{a}_i$ ($\mathbf{dist}_{ij}$) is smaller than from $\mathbf{a}_k$ to $\mathbf{a}_i$ ($\mathbf{dist}_{ik}$). Triplets like this are the input of our metric learning algorithm. By casting the problem as optimization of the inner product of the linear transformation matrix and its transpose, the formulation is based on solving a semidefinite program. The algorithm finds an *optimal* linear transformation that maximizes the margin between distances $\mathbf{dist}_{ij}$ and $\mathbf{dist}_{ik}$.

---

[*]NICTA is funded by the Australian Government as represented by the Department of Broadband, Communications and the Digital Economy and the Australian Research Council through the ICT Center of Excellence program.

A major drawback of this formulation is that current SDP solvers utilizing interior-point (IP) methods do not scale well to large problems with computation complexity roughly $O(n^{4.5})$ ($n$ is the number of variables). On the other hand, linear programming is much better in terms of scalability. State-of-the-art solvers like CPLEX [4] can solve large problems up to millions of variables and constraints. This motivates us to develop an LP approach to solve our SDP metric learning problem.

## 2  Related Work

We overview some relevant work in this section.

Column generation was first proposed by Dantzig and Wolfe [5] for solving some special structured linear programs with extremely large number of variables. [3] has presented a comprehensive survey on this technique. The general idea of CG is that, instead of solving the original large-scale problem (master problem), one works on a restricted master problem with a reasonably small subset of variables at each step. The dual of the restricted master problem is solved by the simplex method, and the optimal dual solution is used to find the new column to be included into the restricted master problem. LPBoost [1] is a direct application of CG in Boosting. For the first time, LPBoost shows that in an LP framework, unknown weak hypotheses can be learned from the dual although the space of all weak hypotheses is infinitely large. This is the highlight of LPBoost, which has directly inspired our work.

Metric learning using convex optimization has attracted a lot of attention recently [6–8]. These work has made it possible to learn distance functions that are more appropriate for a specific task, based on partially labeled data or proximity constraints. These techniques improve classification or clustering accuracy by taking advantage of prior information. There is plenty of work reported. We list a few that are most relevant to ours. [6] learns a Mahalanobis metric for clustering using convex optimization to minimize the distance between examples belonging to the same class, while at the same time restricting examples in difference classes not to be too close. The work in [7] also learns a Mahalanobis metric using SDP by optimizing a modified $k$-NN classifier. They have used first-order alternating projection algorithms, which are faster than generic SDP solvers. The authors in [8] learns a Mahalanobis by considering proximity relationships of training examples. The final formulation is also an SDP. They replace the positive semidefinite (PSD) conic constraint using a sequence of linear constraints under the fact that a diagonal dominance matrix must be PSD (but not *vice versa*). In other words the conic constraint is replaced by a more strict one. The feasibility set shrinks and the solution obtained is not necessarily a solution of the original SDP.

## 3  Preliminaries

We begin with some notational conventions and basic definitions that will be useful.

A bold lower case letter $\mathbf{x}$ represents a column vector and an upper case letter $\mathbf{X}$ is a matrix. We denote the space of $D \times D$ symmetric matrices by $\mathbb{S}^D$, and positive semidefinite matrices by $\mathbb{S}^D_+$. $\mathbf{Tr}(\cdot)$ is the trace of a square matrix and $\langle \mathbf{X}, \mathbf{Z} \rangle = \mathbf{Tr}(\mathbf{X}\mathbf{Z}^\top) = \sum_{ij} \mathbf{X}_{ij}\mathbf{Z}_{ij}$ calculates the inner product of two matrices. An element-wise inequality between two vectors writes $\mathbf{u} \leq \mathbf{v}$, which means $u_i \leq v_i$ for all $i$.

We use $\mathbf{X} \succcurlyeq 0$ to indicate that matrix $\mathbf{X}$ is positive semidefinite. For a matrix $\mathbf{X} \in \mathbb{S}^D$, the following statements are equivalent: (1) $\mathbf{X} \succcurlyeq 0$ ($\mathbf{X} \in \mathbb{S}^D_+$); (2) All eigenvalues of $\mathbf{X}$ are nonnegative ($\lambda_i(\mathbf{X}) \geq 0, i = 1, \cdots, D$); and (3) $\forall \mathbf{u} \in \mathbb{R}^D, \mathbf{u}^\top \mathbf{X} \mathbf{u} \geq 0$.

### 3.1  Extreme Points of Trace-one Semidefinite Matrices

Before we present our main results, we prove an important theorem that serves the basis of the proposed algorithm.

**Definition 3.1** *For any positive integer $M$, given a set of points $\{\mathbf{x}_1, ..., \mathbf{x}_M\}$ in a real vector or matrix space* Sp*, the* convex hull *of* Sp *spanned by $M$ elements in* Sp *is defined as:*

$$\mathbf{conv}_M(\mathrm{Sp}) = \left\{ \sum_{i=1}^{M} \theta_i \mathbf{x}_i \,\middle|\, \theta_i \geq 0, \sum_{i=1}^{M} \theta_i = 1, \mathbf{x}_i \in \mathrm{Sp} \right\}.$$

*Define the convex hull[1] of* Sp *as:*

$$\mathbf{conv}(\mathrm{Sp}) = \bigcup_M \mathbf{conv}_M(\mathrm{Sp})$$

$$= \left\{ \sum_{i=1}^M \theta_i \mathbf{x}_i \,\middle|\, \theta_i \geq 0, \sum_{i=1}^M \theta_i = 1, \mathbf{x}_i \in \mathrm{Sp}, M \in \mathbb{Z}_+ \right\}.$$

*Here $\mathbb{Z}_+$ denotes the set of all positive integers.*

**Definition 3.2** *Let us define $\Gamma_1$ to be the space of all positive semidefinite matrices $\mathbf{X} \in \mathbb{S}_+^D$ with trace equaling one:*

$$\Gamma_1 = \{ \mathbf{X} \,|\, \mathbf{X} \succcurlyeq 0, \mathbf{Tr}(\mathbf{X}) = 1 \} ;\,^2$$

*and $\Omega_1$ to be the space of all positive semidefinite matrices with both trace and rank equaling one:*

$$\Omega_1 = \{ \mathbf{Z} \,|\, \mathbf{Z} \succcurlyeq 0, \mathbf{Tr}(\mathbf{Z}) = 1, \mathbf{rank}(\mathbf{Z}) = 1 \} .$$

*We also define $\Gamma_2$ as the convex hull of $\Omega_1$, i.e.,*

$$\Gamma_2 = \mathbf{conv}(\Omega_1).$$

**Lemma 3.3** *Let $\Omega_2$ be a convex polytope defined as $\Omega_2 = \{ \boldsymbol{\lambda} \in \mathbb{R}^D \,|\, \lambda_k \geq 0, \forall k = 1, \cdots, D,$ $\sum_{k=1}^D \lambda_k = 1 \}$, then the points with only one element equaling one and all the others being zeros are the extreme points (vertexes) of $\Omega_2$. All the other points can not be extreme points.*

**Proof:** Without loss of generality, let us consider such a point $\boldsymbol{\lambda}' = \{1, 0, \cdots, 0\}$. If $\boldsymbol{\lambda}'$ is not an extreme point of $\Omega_2$, then it must be expressed as an convex combination of a few *other* points in $\Omega_2$: $\boldsymbol{\lambda}' = \sum_{i=1}^M \theta_i \boldsymbol{\lambda}^i, \theta_i > 0, \sum_{i=1}^M \theta_i = 1$ and $\boldsymbol{\lambda}^i \neq \boldsymbol{\lambda}'$. Then we have equations: $\sum_{i=1}^M \theta_i \lambda_k^i = 0$, $\forall k = 2, \cdots, D$. It follows that $\lambda_k^i = 0, \forall i$ and $k = 2, \cdots, D$. That means, $\lambda_1^i = 1 \ \forall i$. This is inconsistent with $\boldsymbol{\lambda}^i \neq \boldsymbol{\lambda}'$. Therefore such a convex combination does not exist and $\boldsymbol{\lambda}'$ must be an extreme point. It is trivial to see that any $\boldsymbol{\lambda}$ that has more than one active element is an convex combination of the above-defined extreme points. So they can not be extreme points. □

**Theorem 3.4** *$\Gamma_1$ equals to $\Gamma_2$; i.e., $\Gamma_1$ is also the convex hull of $\Omega_1$. In other words, all $\mathbf{Z} \in \Omega_1$, forms the set of extreme points of $\Gamma_1$.*

**Proof:** It is easy to check that any convex combination $\sum_i \theta_i \mathbf{Z}^i$, such that $\mathbf{Z}^i \in \Omega_1$, resides in $\Gamma_1$, with the following two facts: (1) a convex combination of PSD matrices is still a PSD matrix; (2) $\mathbf{Tr}\left(\sum_i \theta_i \mathbf{Z}^i\right) = \sum_i \left(\theta_i \mathbf{Tr}(\mathbf{Z}^i)\right) = 1$.

By denoting $\lambda_1 \geq \cdots \geq \lambda_D \geq 0$ the eigenvalues of a $\mathbf{Z} \in \Gamma_1$, we know that $\lambda_1 \leq 1$ because $\sum_{i=1}^D \lambda_i = \mathbf{Tr}(\mathbf{Z}) = 1$. Therefore, all eigenvalues of $\mathbf{Z}$ must satisfy: $\lambda_i \in [0, 1], \forall i = 1, \cdots, D$ and $\sum_i^D \lambda_i = 1$. By looking at the eigenvalues of $\mathbf{Z}$ and using Lemma 3.3, it is immediate to see that a matrix $\mathbf{Z}$ such that $\mathbf{Z} \succcurlyeq 0, \mathbf{Tr}(\mathbf{Z}) = 1$ and $\mathbf{rank}(\mathbf{Z}) > 1$ can not be an extreme point of $\Gamma_1$. The only candidates for extreme points are those rank-one matrices ($\lambda_1 = 1$ and $\lambda_{2,\cdots,D} = 0$). Moreover, it is not possible that some rank-one matrices are extreme points and others are not because the other two constraints $\mathbf{Z} \succcurlyeq 0$ and $\mathbf{Tr}(\mathbf{Z}) = 1$ do not distinguish between different rank-one matrices.

Hence, all $\mathbf{Z} \in \Omega_1$ forms the set of extreme points of $\Gamma_1$. Furthermore, $\Gamma_1$ is a convex and compact set, which must have extreme points. Krein-Milman Theorem [9] tells us that a convex and compact set is equal to the convex hull of its extreme points. □

This theorem is a special case of the results from [10] in the context of eigenvalue optimization. A different proof for the above theorem's general version can also be found in [11]. In the context of SDP optimization, what is of interest about Theorem 3.4 is as follows: it tells us that a bounded PSD matrix constraint $\mathbf{X} \in \Gamma_1$ can be equivalently replaced with a set of constrains which belong to $\Gamma_2$. At the first glance, this is a highly counterintuitive proposition because $\Gamma_2$ involves many more complicated constraints. Both $\theta_i$ and $\mathbf{Z}^i$ ($\forall i = 1, \cdots, M$) are unknown variables. Even worse, $M$ could be extremely (or even indefinitely) large.

## 3.2  Boosting

Boosting is an example of ensemble learning, where multiple learners are trained to solve the same problem. Typically a boosting algorithm [12] creates a single strong learner by incrementally adding base (weak) learners to the final strong learner. The base learner has an important impact on the strong learner. In general, a boosting algorithm builds on a user-specified base learning procedure and runs it repeatedly on modified data that are outputs from the previous iterations.

The inputs to a boosting algorithm are a set of training example $\mathbf{x}$, and their corresponding class labels $y$. The final output strong classifier takes the form

$$F_{\boldsymbol{\theta}}(\mathbf{x}) = \sum\nolimits_{i=1}^{M} \theta_i f_i(\mathbf{x}). \tag{1}$$

Here $f_i(\cdot)$ is a base learner. From Theorem 3.4, we know that a matrix $\mathbf{X} \in \Gamma_1$ can be decomposed as

$$\mathbf{X} = \sum\nolimits_{i=1}^{M} \theta_i \mathbf{Z}^i, \mathbf{Z}^i \in \Omega_1. \tag{2}$$

By observing the similarity between Equations (1) and (2), we may view $\mathbf{Z}^i$ as a weak classifier and the matrix $\mathbf{X}$ as the strong classifier we want to learn. This is exactly the problem that boosting methods have been designed to solve. This observation inspires us to solve a special type of SDPs using boosting techniques.

A sparse greedy approximation algorithm proposed by Zhang [2] is an efficient way of solving a class of convex problems, which provides fast convergence rates. It is shown in [2] that boosting algorithms can be interpreted within the general framework of [2]. The main idea of sequential greedy approximation is as follows. Given an initialization $\mathbf{u}^0 \in \mathbb{V}$, $\mathbb{V}$ can be a subset of a linear vector space, a matrix space or a functional space. The algorithm finds $\mathbf{u}^i \in \mathbb{V}$, $i = 1, \cdots$, and $0 \le \lambda \le 1$ such that the cost function $F((1-\lambda)\mathbf{u}^{i-1} + \lambda\mathbf{u}^i)$ is approximately minimized; Then the solution $\mathbf{u}^i$ is updated as $\mathbf{u}^i = (1-\lambda)\mathbf{u}^{i-1} + \lambda\mathbf{u}^i$ and the iteration goes on.

## 4  Large-margin Semidefinite Metric Learning

We consider the Mahalanobis metric learning problem as an example although the proposed technique can be applied to many other problems in machine learning such as nonparametric kernel matrix learning [13].

We are given a set of training examples $\mathbf{a}_i \in \mathbb{R}^D$, $i = 1, 2, \cdots$. The task is to learn a distance metric such that with the learned metric, classification or clustering will achieve better performance on testing data. The information available is a bunch of relative distance comparisons. Mathematically we are given a set $\mathcal{S}$ which contains the training triplets: $\mathcal{S} = \{(\mathbf{a}_i, \mathbf{a}_j, \mathbf{a}_k)| \, \mathbf{dist}_{ij} < \mathbf{dist}_{ik}\}$, where $\mathbf{dist}_{ij}$ measures distance between $\mathbf{a}_i$ and $\mathbf{a}_j$ with a certain metric. In this work we focus on the case that $\mathbf{dist}$ calculates the Mahalanobis distance. Equivalently we are learning a linear transformation $\mathbf{P} \in \mathbb{R}^{D \times d}$ such that $\mathbf{dist}$ is the Euclidean distance in the projected space: $\mathbf{dist}_{ij} = \left\| \mathbf{P}^\top \mathbf{a}_i - \mathbf{P}^\top \mathbf{a}_j \right\|_2^2 = (\mathbf{a}_i - \mathbf{a}_j)^\top \mathbf{P}\mathbf{P}^\top (\mathbf{a}_i - \mathbf{a}_j)$. It is not difficult to see that the inequalities in the set $\mathcal{S}$ are non-convex because a difference of quadratic terms in $\mathbf{P}$ is involved. In order to *convexify* the inequalities in $\mathcal{S}$, a new variable $\mathbf{X} = \mathbf{P}\mathbf{P}^\top$ is instead used. This is a typical technique for modeling an SDP problem [14]. We wish to maximize the margin that is defined as the distance between $\mathbf{dist}_{ij}$ and $\mathbf{dist}_{ik}$. That is, $\rho = \mathbf{dist}_{ik} - \mathbf{dist}_{ij} = (\mathbf{a}_i - \mathbf{a}_k)^\top \mathbf{X}(\mathbf{a}_i - \mathbf{a}_k) - (\mathbf{a}_i - \mathbf{a}_j)^\top \mathbf{X}(\mathbf{a}_i - \mathbf{a}_j)$. Also one may use soft margin to tolerate noisy data. Putting these thoughts together, the final convex program we want to optimize is:

$$
\begin{aligned}
\max_{\rho, \mathbf{X}, \boldsymbol{\xi}} \quad & \rho - C \sum\nolimits_{r=1}^{|\mathcal{S}|} \xi_r \\
\text{s.t.} \quad & \mathbf{X} \succcurlyeq 0, \mathbf{Tr}(\mathbf{X}) = 1, \boldsymbol{\xi} \ge 0, \\
& (\mathbf{a}_i - \mathbf{a}_k)^\top \mathbf{X}(\mathbf{a}_i - \mathbf{a}_k) - (\mathbf{a}_i - \mathbf{a}_j)^\top \mathbf{X}(\mathbf{a}_i - \mathbf{a}_j) \ge \rho - \xi_r, \\
& \forall (\mathbf{a}_i, \mathbf{a}_j, \mathbf{a}_k) \in \mathcal{S}.
\end{aligned}
\tag{3}
$$

Here $r$ indexes the training set $\mathcal{S}$. $|\mathcal{S}|$ denotes the size of $\mathcal{S}$. $C$ is a trade-off parameter that balances the training error and the margin. Same as in support vector machine, the slack variable $\boldsymbol{\xi} \ge 0$

corresponds to the soft-margin hinge loss. Note that the constraint $\mathbf{Tr}(\mathbf{X}) = 1$ removes the scale ambiguity because the distance inequalities are scale invariant.

To simplify our exposition, we write

$$\mathbf{A}^r = (\mathbf{a}_i - \mathbf{a}_k)(\mathbf{a}_i - \mathbf{a}_k)^\top - (\mathbf{a}_i - \mathbf{a}_j)(\mathbf{a}_i - \mathbf{a}_j)^\top. \tag{4}$$

The last constraint in (3) is then written

$$\langle \mathbf{A}^r, \mathbf{X} \rangle \geq \rho - \xi_r, \ \forall \mathbf{A}^r \text{ built from } \mathcal{S}; r = 1, \cdots |\mathcal{S}|. \tag{5}$$

Problem (3) is a typical SDP since it has a linear cost function and linear constraints plus a PSD conic constraint. Therefore it can be solved using off-the-shelf SDP solvers like CSDP [15]. As mentioned general interior-point SDP solvers do not scale well to large-sized problems. Current solvers can only solve problems up to a few thousand variables, which makes many applications intractable. For example, in face recognition if the inputs are $30 \times 30$ images, then $D = 900$ and there would be $0.41$ million variables. Next we show how we reformulate the above SDP into an LP.

## 5   Boosting via Matrix-Generation Linear Programming

Using Theorem 3.4, we can replace the PSD conic constraint in (3) with a linear convex combination of rank-one unitary PSD matrices: $\mathbf{X} = \sum_{i=1}^M \theta_i \mathbf{Z}^i$. Substituting $\mathbf{X}$ in Problem (3), we obtain

$$\max_{\rho,\boldsymbol{\theta},\boldsymbol{\xi},\mathbf{Z}} \ \rho - C \sum_{r=1}^{|\mathcal{S}|} \xi_r$$
$$\text{s.t. } \boldsymbol{\xi} \geq 0,$$
$$\left\langle \mathbf{A}^r, \textstyle\sum_{i=1}^M \theta_i \mathbf{Z}^i \right\rangle = \sum_{i=1}^M \left\langle \mathbf{A}^r, \mathbf{Z}^i \right\rangle \theta_i \geq \rho - \xi_r,$$
$$\forall \mathbf{A}^r \text{ built from } \mathcal{S}; r = 1, \cdots |\mathcal{S}|, \tag{$P_1$}$$
$$\textstyle\sum_{i=1}^M \theta_i = 1, \boldsymbol{\theta} \geq 0,$$
$$\mathbf{Z}^i \in \Omega_1, i = 1, \cdots, M.$$

This above problem is still very hard to solve since it has non-convex rank constraints and an indefinite number of variables ($M$ is indefinite because there are an indefinite number of rank-one matrices). However if we somehow know matrices $\mathbf{Z}^i$ ($i = 1, \cdots$) *a priori*, we can then drop all the constraints imposed on $\mathbf{Z}^i$ ($i = 1, \cdots$) and the problem becomes a linear program; or more precisely a semi-infinite linear program (SILP) because it has an infinitely large set of variables $\boldsymbol{\theta}$.

Column generation is a state-of-the-art method for optimally solving difficult large-scale optimization problems. It is a method to avoid considering all variables of a problem *explicitly*. If an LP has extremely many variables (columns) but much fewer constraints, CG can be very beneficial. The crucial insight behind CG is: for an LP problem with many variables, the number of non-zero variables of the optimal solution is equal to the number of constraints, hence although the number of possible variables may be large, we only need a small subset of these in the optimal solution. It works by only considering a small subset of the entire variable set. Once it is solved, we ask the question: "Are there any other variables that can be included to improve the solution?". So we must be able to solve the *subproblem*: given a set of dual values, one either identifies a variable that has a favorable reduced cost, or indicates that such a variable does not exist. In essence, CG finds the variables with negative reduced costs without explicitly enumerating all variables. For a general LP, this may not be possible. But for some types of problems it is possible.

We now consider Problem ($P_1$) as if all $\mathbf{Z}^i$, ($i = 1, \cdots$) were known. The dual of ($P_1$) is easily derived:

$$\min_{\pi,\boldsymbol{w}} \ \pi$$
$$\text{s.t. } \textstyle\sum_{r=1}^{|\mathcal{S}|} \left\langle \mathbf{A}^r, \mathbf{Z}^i \right\rangle w_r \leq \pi, \ i = 1, \cdots, M, \tag{$D_1$}$$
$$\textstyle\sum_{r=1}^{|\mathcal{S}|} w_r = 1,$$
$$0 \leq w_r \leq C, \ r = 1, \cdots, |\mathcal{S}|.$$

For convex programs with strong duality, the dual gap is zeros, which means the optimal value of the primal and dual problems coincide. For LPs and SDPs, strong duality holds under very mild conditions (almost always satisfied by LPs and SDPs considered here).

We now only consider a small subset of the variables in the primal; *i.e.*, only a subset of $\mathbf{Z}$ (denoted by $\tilde{\mathbf{Z}}$)[3] is used. The LP solved using $\tilde{\mathbf{Z}}$ is usually termed *restricted master problem* (RMP). Because the primal variables correspond to the dual constraints, solving RMP is equivalent to solving a relaxed version of the dual problem. With a finite $\tilde{\mathbf{Z}}$, the first set of constraints in (D$_1$) are finite, and we can solve the LP that satisfies all the existing constraints.

If we can prove that among all the constraints that we have not added to the dual problem, no single constraint is violated, then we can conclude that solving the restricted problem is equivalent to solving the original problem. Otherwise, there exists at least one constraint that is violated. The violated constraints correspond to variables in primal that are not in RMP. Adding these variables to RMP leads to a new RMP that needs to be re-optimized. In our case, by finding the violated constraint, we generate a rank-one matrix $\mathbf{Z}'$. Hence, as in LPBoost [1] we have a base learning algorithm as an oracle that either finds a new $\mathbf{Z}'$ such that

$$\sum_{r=1}^{|\mathcal{S}|} \langle \mathbf{A}^r, \mathbf{Z}' \rangle w_r > \tilde{\pi},$$

where $\tilde{\pi}$ is the solution of the current restricted problem, or a guarantee that such a $\mathbf{Z}'$ does not exist. To make convergence fast, we find the one that has largest deviation. That is,

$$\mathbf{Z}' = \mathrm{argmax}_{\mathbf{Z}} \left\{ \sum_{r=1}^{|\mathcal{S}|} \langle \mathbf{A}^r, \mathbf{Z} \rangle \tilde{w}_r, \text{ s.t. } \mathbf{Z} \in \Omega_1 \right\}. \tag{B$_1$}$$

Again here $\tilde{w}_r$ ($r = 1, \cdots, |\mathcal{S}|$) are obtained by solving the current restricted dual problem (D$_1$). Let us denote $\mathrm{Opt}(\mathrm{B}_1)$ the optimal value of the optimization problem in (B$_1$). We now have a criterion that guarantees the optimal convex combination over all $\mathbf{Z}$'s satisfying the constraints in $\Gamma_2$ has been found. If $\mathrm{Opt}(\mathrm{B}_1) \leq \tilde{\pi}$, then we are done—we have solved the original problem.

The presented algorithm is a variant of the CG technique. At each iteration, a new matrix is generated, hence the name *matrix generation*.

## 5.1 Base Learning Algorithm

In this section, we show that the optimization problem (B$_1$) can be exactly and efficiently solved using eigen-decomposition.

From $\mathbf{Z} \succcurlyeq 0$ and $\mathbf{rank}(\mathbf{Z}) = 1$, we know that $\mathbf{Z}$ has the format: $\mathbf{Z} = \mathbf{u}\mathbf{u}^\top$, $\mathbf{u} \in \mathbb{R}^D$; and $\mathbf{Tr}(\mathbf{Z}) = 1$ means $\|\mathbf{u}\|_2 = 1$. We have

$$\sum_{r=1}^{|\mathcal{S}|} \langle \mathbf{A}^r, \mathbf{Z} \rangle \tilde{w}_r = \left\langle \sum_{r=1}^{|\mathcal{S}|} \tilde{w}_r \mathbf{A}^r, \mathbf{Z} \right\rangle = \mathbf{u} \left( \sum_{r=1}^{|\mathcal{S}|} \tilde{w}_r \mathbf{A}^r \right) \mathbf{u}^\top.$$

By denoting

$$\tilde{\mathbf{H}} = \sum_{r=1}^{|\mathcal{S}|} \tilde{w}_r \mathbf{A}^r, \tag{6}$$

the optimization in (B$_1$) equals:

$$\max_{\mathbf{u}} \ \mathbf{u}^\top \tilde{\mathbf{H}} \mathbf{u}, \text{ subject to } \|\mathbf{u}\|_2 = 1. \tag{7}$$

It is clear that the largest eigenvalue of $\tilde{\mathbf{H}}$, $\lambda_{\max}(\tilde{\mathbf{H}})$, and its corresponding eigenvector $\mathbf{u}_1$ give the solution to the above problem. Note that $\tilde{\mathbf{H}}$ is symmetric. Therefore we have the solution of the original problem (B$_1$): $\mathrm{Opt}(\mathrm{B}_1) = \lambda_{\max}(\tilde{\mathbf{H}})$ and $\mathbf{Z}' = \mathbf{u}_1 \mathbf{u}_1^\top$.

There are approximate eigenvalue solvers, which guarantee that for a symmetric matrix $\mathbf{U}$ and any $\varepsilon > 0$, a vector $\mathbf{v}$ is found such that $\mathbf{v}^\top \mathbf{U} \mathbf{v} \geq \lambda_{\max} - \varepsilon$. To approximately find the largest eigenvalue and eigenvector can be very efficient using Lanczos or power method. We use the MATLAB function *eigs* to calculate the largest eigenvector, which calls mex files of ARPACK. ARPACK is a collection of Fortran subroutines designed to solve large scale eigenvalue problems. When the input matrix is symmetric, this software uses a variant of the Lanczos process called the implicitly restarted Lanczos method [16].

**Algorithm 1**: PSDBoost for semidefinite metric learning.

---

**Input**: Training set triplets $(\mathbf{a}_i, \mathbf{a}_j, \mathbf{a}_k) \in \mathcal{S}$; Calculate $\mathbf{A}^r$, $r = 1, \cdots$ from $\mathcal{S}$ using Equation (4).

**Initialization**:

    1. $M = 1$ (no bases selected);

    2. $\boldsymbol{\theta} = \mathbf{0}$ (all primal coefficients are zeros);

    3. $\pi = 0$;

    4. $w_r = \frac{1}{|\mathcal{S}|}$, $r = 1, \cdots, |\mathcal{S}|$ (uniform dual weights).

**while** true **do**

        1. Find a new base $\mathbf{Z}'$ by solving Problem (B$_1$), *i.e.*, eigen-decomposition of $\tilde{\mathbf{H}}$ in (6);

        2. **if** $\mathrm{Opt}(\mathrm{B}_1) \leq \pi$ **then** break (problem solved);

        3. Add $\mathbf{Z}'$ to the restricted master problem, which corresponds to a new constraint in Problem (D$_1$);

        4. Solve the dual (D$_1$) to obtain updated $\pi$ and $w_r$ ($r = 1, \cdots, |\mathcal{S}|$);

        5. $M = M + 1$ (base count).

**end**

**Output**:

    1. Calculate the primal variable $\boldsymbol{\theta}$ from the optimality conditions and the last solved dual LP;

    2. The learned PSD matrix $\mathbf{X} \in \mathbb{R}^{D \times D}$, $\mathbf{X} = \sum_{i=1}^{M} \theta_i \mathbf{Z}^i$.

---

Putting all the above analysis together, we summarize our PSDBoost algorithm for metric learning in Algorithm 1. Note that, in practice, we can relax the convergence criterion by setting a small positive threshold $\varepsilon' > 0$ in order to obtain a good approximation quickly. Namely the convergence criterion is $\mathrm{Opt}(\mathrm{B}_1) \leq \pi + \varepsilon'$.

The algorithm has some appealing properties. Each iteration the solution is provably better than the preceding one, and has rank at most one larger. Hence after $M$ iterations the algorithm attains a solution with rank at most $M$. The algorithm preserves CG's property that each iteration improves the quality of the solution. The bounded rank follows the fact that $\mathbf{rank}(\mathbf{A} + \mathbf{B}) \leq \mathbf{rank}(\mathbf{A}) + \mathbf{rank}(\mathbf{B})$, $\forall$ matrices $\mathbf{A}$ and $\mathbf{B}$.

An advantage of the proposed PSDBoost algorithm over standard boosting schemes is the totally-corrective weight update in each iteration, which leads faster convergence. The coordinate descent optimization employed by standard boosting algorithms is known to have a slow convergence rate in general. However, the price of this totally-corrective update is obvious. PSDBoost spans the space of the parameter $\mathbf{X}$ incrementally. The computational cost for solving the subproblem grows with the number of linear constraints, which increases by one at each iteration. Also it needs more and more memory to store the generated base learner $\mathbf{Z}^i$ as represented by a series of unit vectors. To alleviate this problem, one can use a *selection and compression mechanism* as the aggregation step of bundle methods [17]. When the size of of the bundle becomes too large, bundle methods select columns to be discarded and the selected information is aggregated into a single one. It can be shown that as long as the aggregated column is introduced in the bundle, the bundle algorithm remains convergent, although different selection of discarded columns may lead to different convergence speeds. See [17] for details.

## 6 Experiments

In the first experiment, we have artificially generated $600$ points in $24$ dimensions. Therefore the learned metric is of size $24 \times 24$. The triplets are obtained in this way: For a point $\mathbf{a}_i$, we find its nearest neighbor in the same class $\mathbf{a}_j$ and its nearest neighbor in the different class $\mathbf{a}_k$. We subsample to have $550$ triplets for training. To show the convergence, we have plotted the optimal values of the dual problem (D$_1$) at each iteration in Figure 1. We see that PSDBoost quickly converges to

the near-optimal solution. We have observed the so-called *tailing-off effect* of CG on large datasets. While a near-optimal solution is approached considerably fast, only little progress per iteration is made close to the optimum. Stabilization techniques have been introduced to partially alleviate this problem [3]. However, approximate solutions are sufficient for most machine learning tasks. Moreover, we usually are not interested in the numerical accuracy of the solution but the test error for many problems such as metric and kernel learning.

The second experiment uses the Pendigits data from the UCI repository that contains handwritten samples of digits 1, 5, 7, 9. The data for each digits are 16-dimensional. $80$ samples for each digit are used for training and $500$ for each digit for testing. The results show that PSDBoost converges quickly and the learned metric is very similar to the results obtained by a standard SDP solver. The classification errors on testing data with a 1-nearest neighbor are identical using the metrics learned by PSDBoost and a standard SDP solver. Both are $1.3\%$.

## 7  Conclusion

We have presented a new boosting algorithm, PSDBoost, for learning a positive semidefinite matrix. In particular, as an example, we use PSDBoost to learn a distance metric for classification. PSDBoost can also be used to learn a kernel matrix, which is of interest in machine learning. We are currently exploring new applications with PSDBoost. Also we want to know what kind of SDP optimization problems can be approximately solved by PSDBoost.

## Footnotes

[1] Strictly speaking, the union of convex hulls may not be a convex hull in general. It is a linear convex span.

[2] Such a matrix $\mathbf{X}$ is called a density matrix, which is one of the main concepts in quantum physics. A density matrix of rank one is called a pure state, and a density matrix of rank higher than one is called a mixed state.

[3]We also use $\tilde{\boldsymbol{\theta}}$, $\tilde{\pi}$ and $\tilde{\boldsymbol{w}}$ *etc.* to denote the solution of the current RMP and its dual.

## References

[1] A. Demiriz, K.P. Bennett, and J. Shawe-Taylor. Linear programming boosting via column generation. *Mach. Learn.*, 46(1-3):225–254, 2002.

[2] T. Zhang. Sequential greedy approximation for certain convex optimization problems. *IEEE Trans. Inf. Theory*, 49(3):682–691, 2003.

[3] M. E. Lübbecke and J. Desrosiers. Selected topics in column generation. *Operation Res.*, 53(6):1007–1023, 2005.

[4] ILOG, Inc. CPLEX 11.1, 2008. http://www.ilog.com/products/cplex/.

[5] G. B. Dantzig and P. Wolfe. Decomposition principle for linear programs. *Operation Res.*, 8(1):101–111, 1960.

[6] E. Xing, A. Ng, M. Jordan, and S. Russell. Distance metric learning, with application to clustering with side-information. In *Proc. Adv. Neural Inf. Process. Syst.* MIT Press, 2002.

[7] K. Q. Weinberger, J. Blitzer, and L. K. Saul. Distance metric learning for large margin nearest neighbor classification. In *Proc. Adv. Neural Inf. Process. Syst.*, pages 1473–1480, 2005.

[8] R. Rosales and G. Fung. Learning sparse metrics via linear programming. In *Proc. ACM Int. Conf. Knowledge Discovery & Data Mining*, pages 367–373, Philadelphia, PA, USA, 2006.

[9] M. Krein and D. Milman. On extreme points of regular convex sets. *Studia Mathematica*, 9:133–138, 1940.

[10] M. L. Overton and R. S. Womersley. On the sum of the largest eigenvalues of a symmetric matrix. *SIAM J. Matrix Anal. Appl.*, 13(1):41–45, 1992.

[11] P. A. Fillmore and J. P. Williams. Some convexity theorems for matrices. *Glasgow Math. Journal*, 12:110–117, 1971.

[12] R. E. Schapire. Theoretical views of boosting and applications. In *Proc. Int. Conf. Algorithmic Learn. Theory*, pages 13–25, London, UK, 1999. Springer-Verlag.

[13] B. Kulis, M. Sustik, and I. Dhillon. Learning low-rank kernel matrices. In *Proc. Int. Conf. Mach. Learn.*, pages 505–512, Pittsburgh, Pennsylvania, 2006.

[14] S. Boyd and L. Vandenberghe. *Convex Optimization*. Cambridge University Press, 2004.

[15] B. Borchers. CSDP, a C library for semidefinite programming. *Optim. Methods and Softw.*, 11(1):613–623, 1999.

[16] D. Calvetti, L. Reichel, and D. C. Sorensen. An implicitly restarted Lanczos method for large symmetric eigenvalue problems. *Elec. Trans. Numer. Anal*, 2:1–21, Mar 1994. http://etna.mcs.kent.edu.

[17] J. F. Bonnans, J. C. Gilbert, C. Lemaréchal, and C. A. Sagastizábal. *Numerical Optimization: Theoretical and Practical Aspects (1st edition)*. Springer-Verlag, Berlin, 2003.

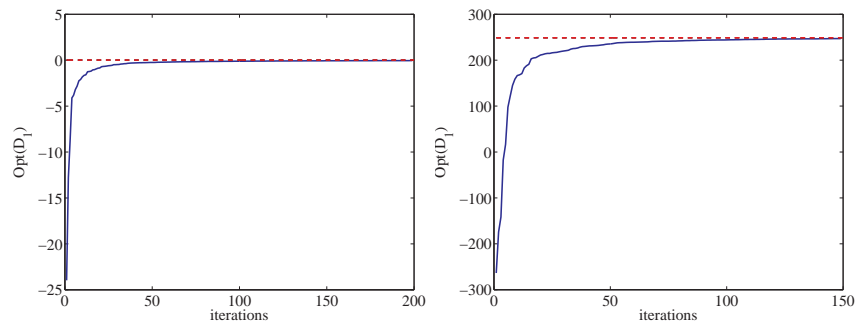

Figure 1: The objective value of the dual problem ($D_1$) on the first (left) and second (right) experiment. The dashed line shows the ground truth obtained by directly solving the original primal SDP (3) using interior-point methods.